# Coding time-varying signals using sparse, shift-invariant representations

**Michael S. Lewicki***        **Terrence J. Sejnowski**
lewicki@salk.edu        terry@salk.edu

Howard Hughes Medical Institute
Computational Neurobiology Laboratory
The Salk Institute
10010 N. Torrey Pines Rd.
La Jolla, CA 92037

## Abstract

A common way to represent a time series is to divide it into short-duration blocks, each of which is then represented by a set of basis functions. A limitation of this approach, however, is that the temporal alignment of the basis functions with the underlying structure in the time series is arbitrary. We present an algorithm for encoding a time series that does not require blocking the data. The algorithm finds an efficient representation by inferring the best temporal positions for functions in a kernel basis. These can have arbitrary temporal extent and are not constrained to be orthogonal. This allows the model to capture structure in the signal that may occur at arbitrary temporal positions and preserves the relative temporal structure of underlying events. The model is shown to be equivalent to a very sparse and highly overcomplete basis. Under this model, the mapping from the data to the representation is nonlinear, but can be computed efficiently. This form also allows the use of existing methods for adapting the basis itself to data. This approach is applied to speech data and results in a shift invariant, spike-like representation that resembles coding in the cochlear nerve.

## 1 Introduction

Time series are often encoded by first dividing the signal into a sequence of blocks. The data within each block is then fit with a standard basis such as a Fourier or wavelet. This has a limitation that the components of the bases are arbitrarily aligned with respect to structure in the time series. Figure 1 shows a short segment of speech data and the boundaries of the blocks. Although the structure in the signal is largely periodic, each large oscillation appears in a different position within the blocks and is sometimes split across blocks. This problem is particularly present for acoustic events with sharp onset, such as plosives in speech. It also presents

difficulties for encoding the signal efficiently, because any basis that is adapted to the underlying structure must represent all possible phases. This can be somewhat circumvented by techniques such as windowing or averaging sliding blocks, but it would be more desirable if the representation were *shift invariant*.

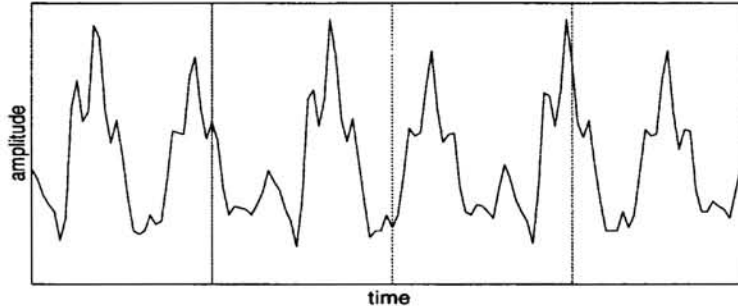

Figure 1: Blocking results in arbitrary phase alignment the underlying structure.

## 2   The Model

Our goal is to model a signal by using a small set of *kernel* functions that can be placed at arbitrary time points. Ultimately, we want to find the minimal set of functions and time points that fit the signal within a given noise level. We expect this type of model to work well for signals composed of events whose onset can occur at arbitrary temporal positions. Examples of these include, musical instruments sounds with sharp attack or plosive sounds in speech.

We assume time series $x(t)$ is modeled by

$$x(t) = \sum_i s_i \phi_{m[i]}(t - \tau_i) + \epsilon(t)\,, \tag{1}$$

where $\tau_i$ indicates the temporal position of the $i^{\text{th}}$ kernel function, $\phi_{m[i]}$, which is scaled by $s_i$. The notation $m[i]$ represents an index function that specifies which of the $M$ kernel functions is present at time $\tau_i$. A single kernel function can occur at multiple times during the time series. Additive noise at time $t$ is given by $\epsilon(t)$.

A more general way to express (1) is to assume that the kernel functions exist at all time points during the signal, and let the non-zero coefficients determine the positions of the kernel functions. In this case, the model can be expressed in convolutional form

$$x(t) \;=\; \sum_m \int s_m(\tau)\phi_m(t-\tau)d\tau + \epsilon(t) \tag{2}$$

$$\;=\; \sum_m s_m(t) * \phi_m(t) + \epsilon(t)\,, \tag{3}$$

where $s_m(\tau)$ is the coefficient at time $\tau$ for kernel function $\phi_m$.

It is also helpful to express the model in matrix form using a discrete sampling of the continuous time series:

$$x = As + \epsilon\,. \tag{4}$$

The basis matrix, $A$, is defined by

$$A = [C(\phi_1)\, C(\phi_2)\, \cdots\, C(\phi_M)]\,, \tag{5}$$

where $C(a)$ is an $N$-by-$N$ circulant matrix parameterized by the vector $a$. This matrix is constructed by replicating the kernel functions at each sample position

$$C(a) = \begin{bmatrix} a_0 & a_{N-1} & \cdots & a_2 & a_1 \\ a_1 & a_0 & \cdots & a_3 & a_2 \\ \cdots & & \cdots & & \cdots \\ a_{N-2} & a_{N-3} & \cdots & a_0 & a_{N-1} \\ a_{N-1} & a_{N-2} & \cdots & a_1 & a_0 \end{bmatrix} \tag{6}$$

The kernels are zero padded to be of length $N$. The length of each kernel is typically much less than the length of the signal, making $A$ very sparse. This can be viewed as a special case of a Toeplitz matrix. Note that the size of $A$ is $MN$-by-$N$, and is thus an example of an overcomplete basis, i.e. a basis with more basis functions than dimensions in the data space (Simoncelli et al., 1992; Coifman and Wickerhauser, 1992; Mallat and Zhang, 1993; Lewicki and Sejnowski, 1998).

## 3   A probabilistic formulation

The optimal coefficient values for a signal are found by maximizing the posterior distribution

$$\hat{s} = \arg\max_s P(s|x, A) = \arg\max_s P(x|A, s)P(s) \tag{7}$$

where $\hat{s}$ is the most probable representation of the signal. Note that omission of the normalizing constant $P(x|A)$ does not change the location of the maximum. This formulation of the problem offers the advantage that the model can fit more general types of distributions and naturally "denoises" the signal. Note that the mapping from $x$ to $\hat{s}$ is *nonlinear* with non-zero additive noise and an overcomplete basis (Chen et al., 1996; Lewicki and Sejnowski, 1998). Optimizing (7) essentially selects out the subset of basis functions that best account for the data.

To define a probabilistic model, we follow previous conventions for linear generative models with additive noise (Cardoso, 1997; Lewicki and Sejnowski, 1998). We assume the noise, $\epsilon$, to have a Gaussian distribution which yields a data likelihood for a given representation of

$$\log P(x|A, s) \propto -\frac{1}{2\sigma^2}(x - As)^2\,. \tag{8}$$

The function $P(s)$ describes the a priori distribution of the coefficients. Under the assumption that $P(s)$ is sparse (highly peaked around zero), maximizing (7) results in very few nonzero coefficients. A compact representation of $\hat{s}$ is to describe the values of the non-zero coefficients and their temporal positions

$$P(s) = \prod_m P(u_m, \tau_m) = \prod_{m=1}^{M} \prod_{i=1}^{n_m} P(u_{m,i})P(\tau_{m,i})\,, \tag{9}$$

where the prior for the non-zero coefficient values, $u_{m,i}$, is assumed to be Laplacian, and the prior for the temporal positions (or intervals), $\tau_{m,i}$, is assumed to be a gamma distribution.

## 4  Finding the best encoding

A difficult challenge presented by the proposed model is finding a computationally tractable method for fitting it to the data. The brute-force approach of generating the basis matrix $A$ generates an intractable number basis functions for signals of any reasonable length, so we need to look for ways of making the optimization of (7) more efficient. The gradient of the log posterior is given by

$$\frac{\partial}{\partial s} \log P(s|A,x) \propto A^T(x - As) + z(s), \tag{10}$$

where $z(s) = (\log P(s))'$. A basic operation required is $v = A^T u$. We saw that $x = As$ can be computed efficiently using convolution (2). Because $A^T$ is also block circulant

$$A^T = \begin{bmatrix} C(\phi_1') \\ \cdots \\ C(\phi_M') \end{bmatrix} \tag{11}$$

where $\phi'(1:N) = \phi(N:-1:1)$. Thus, terms involving $A^T$ can also be computed efficiently using convolution

$$v = A^T u = \begin{bmatrix} \phi_1(-t) * u(t) \\ \cdots \\ \phi_M(-t) * u(t) \end{bmatrix} \tag{12}$$

**Obtaining an initial representation**

An alternative approach to optimizing (7) is to make use of the fact that if the kernel functions are short enough in length, direct multiplication is faster than convolution, and that, for this highly overcomplete basis, most of the coefficients will be zero after being fit to the data. The central problem in encoding the signal then is to determine which coefficients are non-zero, ideally finding a description of the time series with the minimal number of non-zero coefficients. This is equivalent to determining the best set of temporal positions for each of the kernel functions (1).

A crucial step in this approach is to obtain a good initial estimate of the coefficients. One way to do this is to consider the projection of the signal onto each of the basis functions, i.e. $A^T x$. This estimate will be exact (i.e. zero residual error) in the case of zero noise and $A$ orthogonal. For the non-orthogonal, overcomplete case the solution will be approximate, but for certain choices of the basis matrix, an exact representation can still be obtained efficiently (Daubechies, 1990; Simoncelli et al., 1992).

Figure 2 shows examples of convolving two different kernel functions with data. One disadvantage with this initial solution is that the coefficient functions $s_m^0(t)$ are not sparse. For example, even though the signal in figure 2a is composed of only three instances of the kernel function, the convolution is mostly non-zero.

A simple procedure for obtaining a better initial estimate of the most probable coefficients is to select the time locations of the maxima (or extrema) in the convolutions. These are positions where the kernel functions capture the greatest amount of signal structure and where the optimal coefficients are likely to be non-zero. This generates a large number of positions, but their number can be reduced further by selecting only those that contribute significantly, i.e. where the average power is greater than some fraction of the noise level. From these, a basis for the entire signal is constructed by replicating the kernel functions at the appropriate time positions.

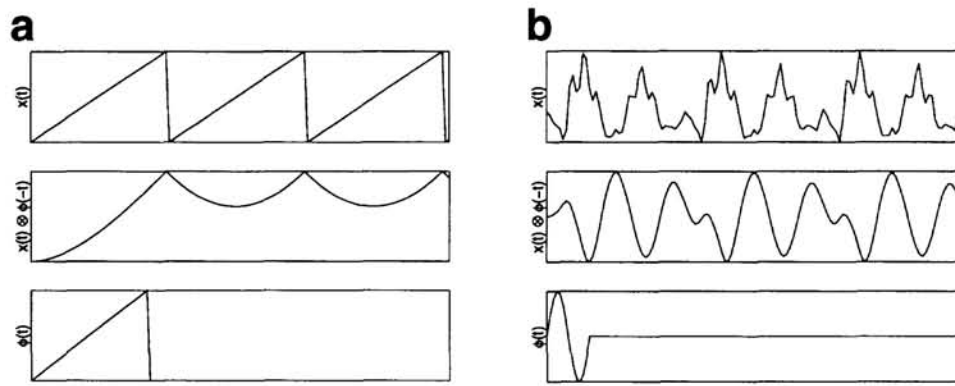

Figure 2: Convolution using the fast Fourier transform is an efficient way to select an initial solution for the temporal positions of the kernel functions. (a) The convolution of a sawtooth-shaped kernel function, $\phi(t)$, with a sawtooth waveform, $x(t)$. (b) A single period sine-wave kernel function convolved with a speech segment.

Once an initial estimate and basis are formed, the most probable coefficient values are estimated using a modified conjugate gradient procedure. The size of the generated basis does not pose a problem for optimization, because it is has very few non-zero elements (the number of which is roughly constant per unit time). This arises because each column is non-zero only around the position of the kernel function, which is typically much shorter in duration than the data waveform. This structure affords the use of sparse matrix routines for all the key computations in the conjugate gradient routine. After the initial fit, there typically are a large number of basis functions that give a very small contribution. These can be pruned to yield, after refitting, a more probable representation that has significantly fewer coefficients.

## 5   Properties of the representation

Figure 3 shows the results of fitting a segment of speech with a sine wave kernel. The 64 kernel functions were constructed using a single period of a sine function whose log frequencies were evenly distributed between 0 and Nyquist (4 kHz), which yielded kernel functions that were minimally correlated (they are not orthogonal because each has only one cycle and is zero elsewhere). The kernel function lengths varied between 2 and 64 samples. The plots show the positions of the non-zero coefficients superimposed on the waveform. The residual errors curves from the fitted waveforms are shown offset, below each waveform. The right axes indicate the kernel function number which increase with frequency. The dots show the starting position of the kernels with non-zero coefficients, with the dot size scaled according to the mean power contribution. This plot is essentially a time/frequency analysis, similar to a wavelet decomposition, but on a finer temporal scale.

Figure 3a shows that the structure in the coefficients repeats for each oscillation in the waveform. Adding a delay leaves the relative temporal structure of the non-zero coefficients mostly unchanged (figure 3b). The small variations between the two sets of coefficients are due to variations in the fitting of the small-magnitude coefficients. Representing the signal in figure 3b with a standard complete basis would result in a very different representation.

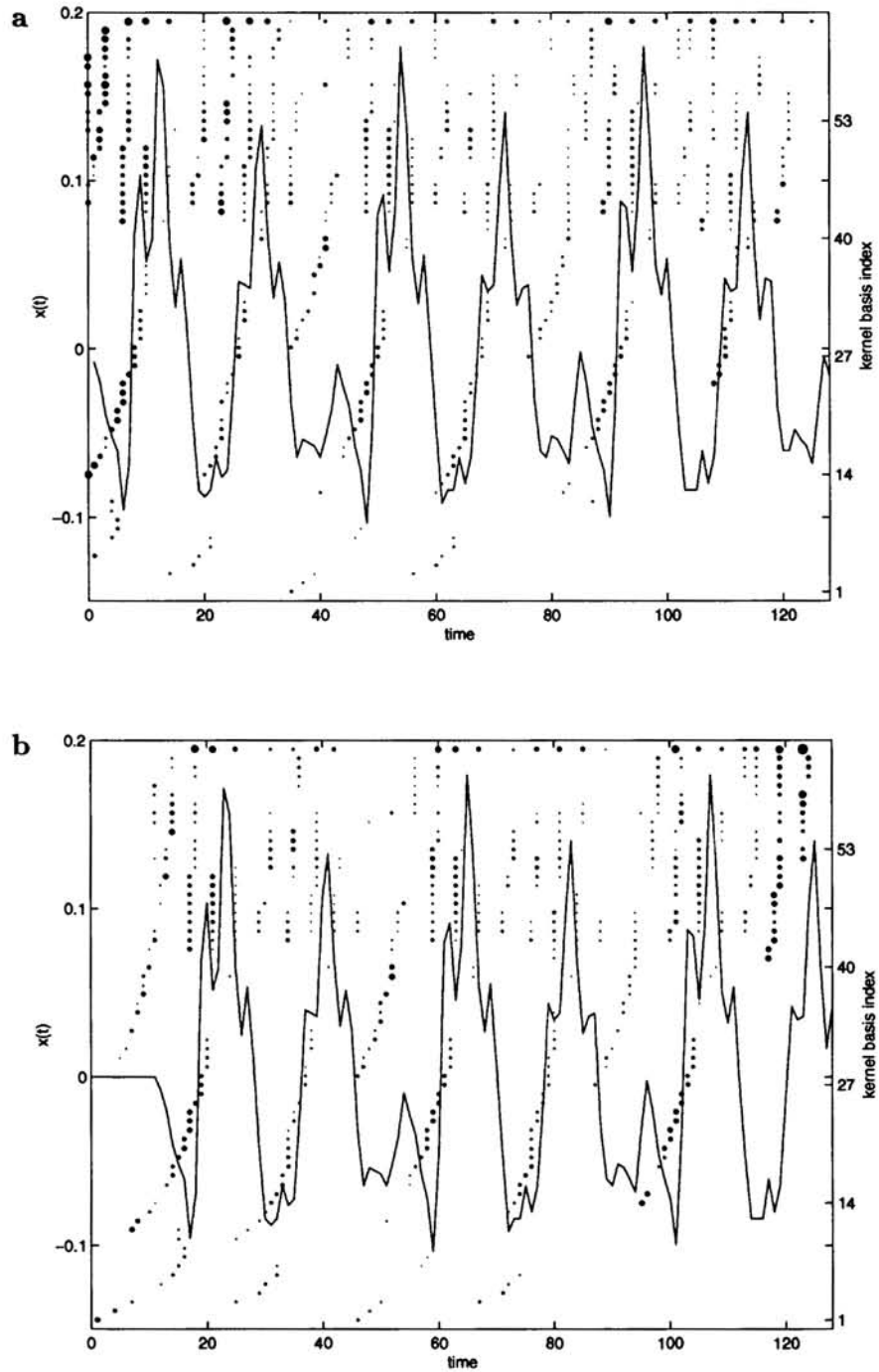

Figure 3: Fitting a shift-invariant model to a segment of speech, x(t). Dots indicate positions of kernels (right axis) with size scaled by the mean power contribution. Fitting error is plotted below speech signal.

## 6   Discussion

The model presented here can be viewed as an extension of the shiftable transforms of Simoncelli et al. (1992). One difference is that here no constraints are placed on the kernel functions. Furthermore, this model accounts for additive noise, which yields automatic signal denoising and provides sensible criteria for selecting significant coefficients. An important unresolved issue is how well the algorithm works for increasingly non-orthogonal kernels.

One interesting property of this representation is that it results in a spike-like representation. In the resulting set of non-zero coefficients, not only is their value important for representing the signal, but also their relative temporal position, which indicate when an underlying event has occurred. This shares many properties with cochlear models. The model described here also has capacity to have an overcomplete representation at any given timepoint, e.g. a kernel basis with an arbitrarily large number of frequencies. These properties make this model potentially useful for binaural signal processing applications.

The effectiveness of this method for efficient coding remains to be proved. A trivial example of a shift-invariant basis is a delta-function model. For a model to encode information efficiently, the representation should be non-redundant. Each basis function should "grab" as much structure in the data as possible and achieve the same level of coding efficiency for arbitrary shifts of the data. The matrix form of the model (4) suggests that it is possible to achieve this optimum by adapting the kernel functions themselves using the methods of Lewicki and Sejnowski (1998). Initial results suggest that this approach is promising. Beyond this, it is evident that modeling the higher-order structure in the coefficients themselves will be necessary both to achieve an efficient representation and to capture structure that is relevant to such tasks as speech recognition or auditory stream segmentation. These results are a step toward these goals.

**Acknowledgments.** We thank Tony Bell, Bruno Olshausen, and David Donoho for helpful discussions.

## Footnotes

*To whom correspondence should be addressed.

## References

Cardoso, J.-F. (1997). Infomax and maximum likelihood for blind source separation. *IEEE Signal Processing Letters*, 4:109–111.

Chen, S., Donoho, D. L., and Saunders, M. A. (1996). Atomic decomposition by basis pursuit. Technical report, Dept. Stat., Stanford Univ., Stanford, CA.

Coifman, R. R. and Wickerhauser, M. V. (1992). Entropy-based algorithms for best basis selection. *IEEE Transactions on Information Theory*, 38(2):713–718.

Daubechies, I. (1990). The wavelet transform, time-frequency localization, and signal analysis. *IEEE Transactions on Information Theory*, 36(5):961–1004.

Lewicki, M. S. and Sejnowski, T. J. (1998). Learning overcomplete representations. *Neural Computation*. submitted.

Mallat, S. G. and Zhang, Z. F. (1993). Matching pursuits with time-frequency dictionaries. *IEEE Transactions on Signal Processing*, 41(12):3397–3415.

Simoncelli, E. P., Freeman, W. T., Adelson, E. H., and J., H. D. (1992). Shiftable multiscale transforms. *IEEE Trans. Info. Theory*, 38:587–607.